# Convergent Fitted Value Iteration
# with Linear Function Approximation

**Daniel J. Lizotte**
David R. Cheriton School of Computer Science
University of Waterloo
Waterloo, ON  N2L 3G1 Canada
dlizotte@uwaterloo.ca

## Abstract

Fitted value iteration (FVI) with ordinary least squares regression is known to diverge. We present a new method, "Expansion-Constrained Ordinary Least Squares" (ECOLS), that produces a linear approximation but also guarantees convergence when used with FVI. To ensure convergence, we constrain the least squares regression operator to be a non-expansion in the $\infty$-norm. We show that the space of function approximators that satisfy this constraint is more rich than the space of "averagers," we prove a minimax property of the ECOLS residual error, and we give an efficient algorithm for computing the coefficients of ECOLS based on constraint generation. We illustrate the algorithmic convergence of FVI with ECOLS in a suite of experiments, and discuss its properties.

## 1  Introduction

Fitted value iteration (FVI), both in the model-based [4] and model-free [5, 15, 16, 17] settings, has become a method of choice for various applied batch reinforcement learning problems. However, it is known that depending on the function approximation scheme used, fitted value iteration can and does diverge in some settings. This is particularly problematic—and easy to illustrate—when using linear regression as the function approximator. The problem of divergence in FVI has been clearly illustrated in several settings [2, 4, 8, 22]. Gordon [8] proved that the class of *averagers*–a very smooth class of function approximators–can safely be used with FVI. Further interest in batch RL methods then led to work that uses non-parametric function approximators with FVI to avoid divergence [5, 15, 16, 17]. This has left a gap in the "middle ground" of function approximator choices that guarantee convergence–we would like to have a function approximator that is more flexible than the averagers but more easily interpreted than the non-parametric approximators. In many scientific applications, linear regression is a natural choice because of its simplicity and interpretability when used with a small set of scientifically meaningful state features. For example, in a medical setting, one may want to base a value function on patient features that are hypothesized to impact a long-term clinical outcome [19]. This enables scientists to interpret the parameters of an optimal learned value function as evidence for or against the importance of these features. Thus for this work, we restrict our attention to linear function approximation, and ensure algorithmic convergence to a fixed point regardless of the generative model of the data. This is in contrast to previous work that explores how properties of the underlying MDP and properties of the function approximation space jointly influence convergence of the algorithm [1, 14, 6].

Our aim is to develop a variant of linear regression that, when used in a fitted value iteration algorithm, guarantees convergence of the algorithm to a fixed point. The contributions of this paper are three-fold: 1) We develop and describe the "Expansion-Constrained Ordinary Least Squares" (ECOLS) approximator. Our approach is to constrain the regression operator to be a non-expansion in the $\infty$-norm. We show that the space of function approximators that satisfy this property is more

rich than the space of averagers [8], and we prove a minimax property on the residual error of the approximator. 2) We give an efficient algorithm for computing the coefficients of ECOLS based on quadratic programming with constraint generation. 3) We verify the algorithmic convergence of fitted value iteration with ECOLS in a suite of experiments and discuss its performance. Finally, we discuss future directions of research and comment on the general problem of learning an interpretable value function and policy from fitted value iteration.

## 2 Background

Consider a finite MDP with states $\mathcal{S} = \{1, ..., n\}$, actions $\mathcal{A} = \{1, ..., |\mathcal{A}|\}$, state transition matrices $P^{(a)} \in \mathbb{R}^{n \times n}$ for each action, a deterministic[1] reward vector $r \in \mathbb{R}^n$, and a discount factor $\gamma < 1$. Let $M_{i,:}$ ($M_{:,i}$) denote the $i$th row (column) of a matrix $M$. The "Bellman optimality" operator or "Dynamic Programming" operator $T$ is given by

$$(Tv)_i = r_i + \max_a \left[ \gamma P_{i,:}^{(a)} v \right]. \tag{1}$$

The fixed point of $T$ is the optimal value function $v^*$ which satisfies the Bellman equation, $Tv^* = v^*$ [3]. From $v^*$ we can recover a policy $\pi_i^* = r_i + \mathrm{argmax}_a \, \gamma P_{i,:}^{(a)} v^*$ that has $v^*$ as its value function. An analogous operator $K$ can be defined for the state-action value function $Q \in \mathbb{R}^{n \times |\mathcal{A}|}$.

$$(KQ)_{i,j} = r_i + \gamma P_{i,:}^{(j)} \max_a Q_{:,a} \tag{2}$$

The fixed point of $K$ is the optimal state-action value $Q^*$ which satisfies $KQ^* = Q^*$. The *value iteration* algorithm proceeds by starting with an initial $v$ or $Q$, and applying $T$ or $K$ repeatedly until convergence, which is guaranteed because both $T$ and $K$ are *contraction mappings* in the infinity norm [8], as we discuss further below. The above operators assume knowledge of the transition model $P^{(a)}$ and rewards $r$. However $K$ in particular is easily adapted to the case of a batch of $n$ tuples of the form $(s_i, a_i, r_i, s_i')$ obtained by interaction with the system [5, 15, 16, 17]. In this case, $Q$ is only evaluated at states in our data set, and in MDPs with continuous state, the number of tuples $n$ is analogous from a computational point of view to the size of our state space.

*Fitted value iteration* [5, 15, 16, 17] (FVI) interleaves either $T$ or $K$ above with a function approximation operator $M$. For example in the model-based case, the composed operator $(M \circ T)$ is applied repeatedly to an initial guess $v^0$. FVI has become increasingly popular especially in the field of "batch-mode Reinforcement Learning" [13, 7] where a policy is learned from a fixed batch of data that was collected by a prior agent. This has particular significance in scientific and medical applications, where ethics concerns prevent the use of current RL methods to interact directly with a trial subject. In these settings, data gathered from controlled trials can still be used to learn good policies [11, 19]. Convergence of FVI depends on properties of $M$—particularly on whether $M$ is a non-expansion in the $\infty$-norm, as we discuss below. The main advantage of fitted value iteration is that the computation of $(M \circ T)$ can be much lower than $n$ in cases where the approximator $M$ only requires computation of elements of $(Tv)_i$ for a small subset of the state space. If $M$ generalizes well, this enables learning in large finite or continuous state spaces. Another advantage is that $M$ can be chosen to represent the value function in a meaningful way, i.e. in a way that meaningfully relates state variables to expected performance. For example, if $M$ were linear regression and a particular state feature had a positive coefficient in the learned value function, we know that larger values of that state feature are preferable. Linear models are of importance because of their ease of interpretation, but unfortunately, ordinary least squares (OLS) function approximation can cause the successive iterations of FVI to fail to converge. We now examine properties of the approximation operator $M$ that control the algorithmic convergence of FVI.

## 3 Non-Expansions and Operator Norms

We say $M$ is a *linear operator* if $My + My' = M(y + y') \ \ \forall y, y' \in \mathbb{R}^p$ and $M0 = 0$. Any linear operator can be represented by a $p \times p$ matrix of real numbers.

By definition, an operator $M$ is a $\gamma$-*contraction* in the $q$-norm if

$$\exists \gamma \leq 1 \text{ s.t. } ||My - My'||_q \leq \gamma ||y - y'||_q \;\; \forall y, y' \in \mathbb{R}^p \tag{3}$$

If the condition holds only for $\gamma = 1$ then $M$ is called a *non-expansion* in the $q$-norm. It is well-known [3, 5, 21] that the operators $T$ and $K$ are $\gamma$-contractions in the $\infty$-norm.

The *operator norm* of $M$ induced by the $q$-norm can be defined in several ways, including

$$||M||_{\text{op}(q)} = \sup_{y \in \mathbb{R}^p, y \neq 0} \frac{||My||_q}{||y||_q}. \tag{4}$$

**Lemma 1.** *A linear operator $M$ is a $\gamma$-contraction in the $q$-norm if and only if $||M||_{\text{op}(q)} \leq \gamma$.*

*Proof.* If $M$ is linear and is a $\gamma$-contraction, we have

$$||M(y - y')||_q \leq \gamma ||y - y'||_q \;\; \forall y, y' \in \mathbb{R}^p. \tag{5}$$

By choosing $y' = 0$, it follows that $M$ satisfies

$$||Mz||_q \leq \gamma ||z||_q \;\; \forall z \in \mathbb{R}^p. \tag{6}$$

Using the definition of $|| \cdot ||_{\text{op}(q)}$, we have that the following conditions are equivalent:

$$||Mz||_q \leq \gamma ||z||_q \;\; \forall z \in \mathbb{R}^p \tag{7}$$

$$\frac{||Mz||_q}{||z||_q} \leq \gamma \;\; \forall z \in \mathbb{R}^p, z \neq 0 \tag{8}$$

$$\sup_{z \in \mathbb{R}^p, z \neq 0} \frac{||Mz||_q}{||z||_q} \leq \gamma \tag{9}$$

$$||M||_{\text{op}(q)} \leq \gamma. \tag{10}$$

Conversely, any $M$ that satisfies (10) satisfies (5) because we can always write $y - y' = z$. $\square$

Lemma 1 implies that a linear operator $M$ is a non-expansion in the $\infty$-norm only if

$$||M||_{\text{op}(\infty)} \leq 1 \tag{11}$$

which is equivalent [18] to:

$$\max_i \sum_j |m_{ij}| \leq 1 \tag{12}$$

**Corollary 1.** *The set of all linear operators that satisfy (12) is exactly the set of linear operators that are non-expansions in the $\infty$-norm.*

One subset of operators on $\mathbb{R}^p$ that are guaranteed to be non-expansions in the $\infty$-norm are the *averagers*[2], as defined by Gordon [8].

**Corollary 2.** *The set of all linear operators that satisfy (12) is larger than the set of averagers.*

*Proof.* For $M$ to be an averager, it must satisfy

$$m_{ij} \geq 0 \; \forall i, j \tag{13}$$

$$\max_i \sum_j m_{ij} \leq 1. \tag{14}$$

These constraints are stricter than (12), because they impose an additional non-negativity constraint on the elements of $M$. $\square$

We have shown that restricting $M$ to be a non-expansion is equivalent to imposing the constraint $||M||_{\text{op}(\infty)} \leq 1$. It is well-known [8] that if such an $M$ is used as a function approximator in fitted value iteration, the algorithm is guaranteed to converge from any starting point because the composition $M \circ T$ is a $\gamma$-contraction in the $\infty$-norm.

## 4 Expansion-Constrained Ordinary Least Squares

We now describe our Expansion-Constrained Ordinary Least Squares function approximation method, and show how we enforce that it is a non-expansion in the $\infty$-norm.

Suppose $X$ is an $n \times p$ design matrix with $n > p$ and $\text{rank}(X) = p$, and suppose $y$ is a vector of regression targets. The usual OLS estimate $\hat{\beta}$ for the model $y \approx X\beta$ is given by

$$\hat{\beta} = \underset{\beta}{\text{argmin}} \, ||X\beta - y||_2 \tag{15}$$

$$= (X^{\mathsf{T}}X)^{-1}X^{\mathsf{T}}y. \tag{16}$$

The predictions made by the model at the points in $X$—i.e., the estimates of $y$—are given by

$$\hat{y} = X\hat{\beta} = X(X^{\mathsf{T}}X)^{-1}X^{\mathsf{T}}y = Hy \tag{17}$$

where $H$ is the "hat" matrix because it "puts the hat" on $y$. The $i$th element of $\hat{y}$ is a linear combination of the elements of $y$, with weights given by the $i$th row of $H$. These weights sum to one, and may be positive or negative. Note that $H$ is a projection of $y$ onto the column space of $X$, and has 1 as an eigenvalue with multiplicity $\text{rank}(X)$, and 0 as an eigenvalue with multiplicity $(n - \text{rank}(X))$.

It is known [18] that for a linear operator $M$, $||M||_{\text{op}(2)}$ is given by the largest singular value of $M$. It follows that $||H||_{\text{op}(2)} \leq 1$ and, by Lemma 1, $H$ is a non-expansion in the 2-norm. However, depending on the data $X$, we may not have $||H||_{\text{op}(\infty)} \leq 1$, in which case $H$ will not be a non-expansion in the $\infty$-norm. The $\infty$-norm expansion property of $H$ is problematic when using linear function approximation for fitted value iteration, as we described earlier.

If one wants to use linear regression safely within a value-iteration algorithm, it is natural to consider constraining the least-squares problem so that the resulting hat matrix is an $\infty$-norm non-expansion. Consider the following optimization problem:

$$\bar{W} = \underset{W}{\text{argmin}} \, ||XWX^{\mathsf{T}}y - y||_2 \tag{18}$$
$$\text{s.t. } ||XWX^{\mathsf{T}}||_{\text{op}(\infty)} \leq 1, \, W \in \mathbb{R}^{p \times p}, W = W^{\mathsf{T}}.$$

The symmetric matrix $W$ is of size $p \times p$, so we have a quadratic objective with a convex norm constraint on $XWX^{\mathsf{T}}$, resulting in a hat matrix $\bar{H} = XWX^{\mathsf{T}}$. If the problem were unconstrained, we would have $\bar{W} = (X^{\mathsf{T}}X)^{-1}$, $\bar{H} = H$ and $\bar{\beta} = \bar{W}X^{\mathsf{T}}y = \hat{\beta}$, the original OLS parameter estimate.

The matrix $\bar{H}$ is a non-expansion by construction. However, unlike the OLS hat matrix $H = X(X^{\mathsf{T}}X)^{-1}X^{\mathsf{T}}$, *the matrix $\bar{H}$ depends on the targets $y$*. That is, given a different set of regression targets, we would compute a different $\bar{H}$. We should therefore more properly write this non-linear operator as $\bar{H}_y$. Because of the non-linearity, the operator $\bar{H}_y$ resulting from the minimization in (18) can in fact be an expansion in the $\infty$-norm despite the constraints.

We now show how we might remove the dependence on $y$ from (18) so that the resulting operator is a linear non-expansion in the $\text{op}(\infty)$-norm. Consider the following optimization problem:

$$\breve{W} = \underset{W}{\text{argmin}} \, \underset{z}{\max} \, ||XWX^{\mathsf{T}}z - z||_2 \tag{19}$$
$$\text{s.t. } ||XWX^{\mathsf{T}}||_{\text{op}(\infty)} \leq 1, \, ||z||_2 = c, \, W \in \mathbb{R}^{p \times p}, W = W^{\mathsf{T}}, \, z \in \mathbb{R}^n$$

Intuitively, the resulting $\breve{W}$ is a linear operator of the form $X\breve{W}X^{\mathsf{T}}$ that minimizes the squared error between its approximation $\breve{z}$ and the *worst-case* (bounded) targets $z$.[3] The resulting $\breve{W}$ does not depend on the regression targets $y$, so the corresponding $\breve{H}$ is a linear operator. The constraint $||XWX^{\mathsf{T}}||_{\text{op}(\infty)} \leq 1$ is effectively a regularizer on the coefficients of the hat matrix which will tend to shrink the fitted values $X\breve{W}X^{\mathsf{T}}y$ toward zero.

Minimization 19 gives us a linear operator, but, as we now show, $\breve{W}$ is not unique—there are in fact an uncountable number of $\breve{W}$ that minimize (19).

**Theorem 1.** *Suppose $W'$ is feasible for (19) and is positive semi-definite. Then $W'$ satisfies*

$$\max_{z,||z||_2<c} ||XW'X^\mathsf{T}z - z||_2 = \min_W \max_{z,||z||_2<c} ||XWX^\mathsf{T}z - z||_2 \tag{20}$$

*for all c.*

*Proof.* We begin by re-formulating (19), which contains a non-concave maximization, as a convex minimization problem with convex constraints.

**Lemma 2.** *Let $X$, $W$, $c$, and $H$ be defined as above. Then*

$$\max_{z,||z||_2=c} ||XWX^\mathsf{T}z - z||_2 = c||XWX^\mathsf{T} - I||_{\text{op}(2)}.$$

*Proof.* $\max_{z\in\mathbb{R}^n,||z||_2=c} ||XWX^\mathsf{T}z - Iz||_2 = \max_{z\in\mathbb{R}^n,||z||_2\leq 1} ||(XWX^\mathsf{T} - I)cz||_2 = c\max_{z\in\mathbb{R}^n,||z||_2\neq 0} ||(XWX^\mathsf{T} - I)z||_2/||z||_2 = c||XWX^\mathsf{T} - I||_{\text{op}(2)}.$
$\square$

Using Lemma 2, we can rewrite (19) as

$$\check{W} = \operatorname*{argmin}_W ||XWX^\mathsf{T} - I||_{\text{op}(2)} \tag{21}$$
$$\text{s.t. } ||XWX^\mathsf{T}||_{\text{op}(\infty)} \leq 1, \ W \in \mathbb{R}^{p\times p}, W = W^\mathsf{T}$$

which is independent of $z$ and independent of the positive constant $c$. This objective is convex in $W$, as are the constraints. We now prove a lower bound on (21) and prove that $W'$ meets the lower bound.

**Lemma 3.** *For all $n\times p$ design matrices $X$ s.t. $n > p$ and all symmetric $W$, $||XWX^\mathsf{T} - I||_{\text{op}(2)} \geq 1$.*

*Proof.* Recall that $||XWX^\mathsf{T} - I||_{\text{op}(2)}$ is given by the largest singular value of $XWX^\mathsf{T} - I$. By symmetry of $W$, write $XWX^\mathsf{T} = UDU^\mathsf{T}$ where $D$ is a diagonal matrix whose diagonal entries $d_{ii}$ are the eigenvalues of $XWX^\mathsf{T}$ and $U$ is an orthonormal matrix. We therefore have

$$XWX^\mathsf{T} - I = UDU^\mathsf{T} - I = UDU^\mathsf{T} - UIU^\mathsf{T} = U(D - I)U^\mathsf{T} \tag{22}$$

Therefore $||XWX^\mathsf{T} - I||_{\text{op}(2)} = \max_i |d_{ii} - 1|$, which is the largest singular value of $XWX^\mathsf{T} - I$. Furthermore we know that $\text{rank}(XWX^\mathsf{T}) \leq p$ and that therefore at least $n - p$ of the $d_{ii}$ are zero. Therefore $\max_i |d_{ii} - 1| \geq 1$, implying $||XWX^\mathsf{T} - I||_{\text{op}(2)} \geq 1$. $\square$

**Lemma 4.** *For any symmetric positive definite matrix $W'$ that satisfies the constraints in (19) and any $n \times p$ design matrix $X$ s.t. $n > p$, we have $||XW'X^\mathsf{T} - I||_{\text{op}(2)} = 1$.*

*Proof.* Let $H' = XW'X^\mathsf{T}$ and write $H' - I = U'(D' - I)U'^\mathsf{T}$ where $U$ is orthogonal and $D'$ is a diagonal matrix whose diagonal entries $d'_{ii}$ are the eigenvalues of $H'$. We know $H'$ is positive semi-definite because $W'$ is assumed to be positive semi-definite; therefore $d'_{ii} \geq 0$. From the constraints in (19), we have $||H'||_{\text{op}(\infty)} \leq 1$, and by symmetry of $H'$ we have $||H'||_{\text{op}(\infty)} = ||H'||_{\text{op}(1)}$. It is known [18] that for any $M$, $||M||_{\text{op}(2)} \leq \sqrt{||M||_{\text{op}(\infty)}||M||_{\text{op}(1)}}$ which gives $||H'||_{\text{op}(2)} \leq 1$ and therefore $|d'_{ii}| \leq 1$ for all $i \in 1..n$. Combining these results gives $0 \leq d'_{ii} \leq 1 \ \forall i$. Recall that $||XW'X^\mathsf{T} - I||_{\text{op}(2)} = \max_i |d_{ii} - 1|$, the maximum eigenvalue of $H'$. Because $\text{rank}(XWX^\mathsf{T}) \leq p$, we know that there exists an $i$ such that $d'_{ii} = 0$, and because we have shown that $0 \leq d'_{ii} \leq 1$, it follows that $\max_i |d_{ii} - 1| = 1$, and therefore $||XW'X^\mathsf{T} - I||_{\text{op}(2)} = 1$. $\square$

Lemma 4 shows that the objective value at any feasible, symmetric postive-definite $W'$ matches the lower bound proved in Lemma 3, and that therefore any such $W'$ satisfies the theorem statement. $\square$

Theorem 1 shows that the optimum of (19) not unique. We therefore solve the following optimization problem, which has a unique solution, shows good empirical performance, and yet still provides the minimax property guaranteed by Theorem 1 when the optimal matrix is positive semi-definite.[4]

$$\tilde{W} = \operatorname*{argmin}_{W} \max_{z} ||XWX^{\mathsf{T}}z - Hz||_2 \tag{23}$$

$$\text{s.t. } ||XWX^{\mathsf{T}}||_{\mathrm{op}(\infty)} \leq 1, \ ||z||_2 = c, \ W \in \mathbb{R}^{p \times p}, W = W^{\mathsf{T}}, \ z \in \mathbb{R}^n$$

Intuitively, this objective searches for a $\tilde{W}$ such that linear approximation using $X\tilde{W}^{\mathsf{T}}X^{\mathsf{T}}$ is as close as possible to the OLS approximation, for the worst case regression targets, according to the 2-norm.

## 5 Computational Formulation

By an argument identical to that of Lemma 2, we can re-formulate (23) as a convex optimization problem with convex constraints, giving

$$\tilde{W} = \operatorname*{argmin}_{W} ||XWX^{\mathsf{T}} - H||_{\mathrm{op}(2)} \tag{24}$$

$$\text{s.t. } ||XWX^{\mathsf{T}}||_{\mathrm{op}(\infty)} \leq 1, \ W \in \mathbb{R}^{p \times p}, W = W^{\mathsf{T}}.$$

Though convex, objective (24) has no simple closed form, and we found that standard solvers have difficulty for larger problems [9]. However, $||XWX^{\mathsf{T}} - H||_{\mathrm{op}(2)}$ is upper bounded by the Frobenius norm $||M||_F = (\sum_{i,j} m_{ij}^2)^{1/2}$. Therefore, we minimize the quadratic objective $||XWX^{\mathsf{T}} - H||_F$ subject to the same convex constraints, which is easier to solve than (21). Note that Theorem 1 applies to the solution of this modified objective when the resulting $\tilde{W}$ is positive semidefinite. Expanding $||XWX^{\mathsf{T}} - H||_F$ gives $||XWX^{\mathsf{T}} - H||_F = \operatorname{Tr}\left[XWX^{\mathsf{T}}XWX^{\mathsf{T}} - 2XWX^{\mathsf{T}} - H\right]$. Let $M(:)$ be the length $p \cdot n$ vector consisting of the stacked columns of the matrix $M$. After some algebraic manipulations, we can re-write the objective as $W(:)^{\mathsf{T}}\Xi W(:) - 2\zeta^{\mathsf{T}}W(:)$, where $\Xi = \sum_{i=1}^n \sum_{j=1}^n \xi^{(ij)}\xi^{(ij)\mathsf{T}}$ and $\xi^{(ij)} = (X_{i,:}^{\mathsf{T}}X_{j,:})(:)$, and $\zeta = (X^{\mathsf{T}}X)(:)$. This objective can then be fed into any standard QP solver. The constraint $||XWX^{\mathsf{T}}||_{\mathrm{op}(\infty)} \leq 1$ can be expressed as the set of constraints $\sum_{j=1}^n |X_{i,:}WX_{j,:}^{\mathsf{T}}| < 1$, $i = 1..n$, or as a set of $n2^n$ linear constraints $\sum_{j=1}^n k_j X_{i,:}WX_{j,:}^{\mathsf{T}} < 1$, $i = 1..n, k \in \{+1, -1\}^n$. Each of these linear constraints involves a vector $k$ with entries $\{+1, -1\}$ multiplied by a row of $XWX^{\mathsf{T}}$. If the entries in $k$ match the signs of the row of $XWX^{\mathsf{T}}$, then their inner product is equal to the sum of the absolute values of the row, which must be constrained. If they do not match, the result is smaller. By constraining all $n2^n$ patterns of signs, we constrain the sum of the absolute values of the entries in the row. Explicitly enforcing all of these constraints is intractable, so we employ a constraint-generation approach [20]. We solve a sequence of quadratic programs, adding the most violated linear constraint after each step. The most violated constraint is given by a row $i^* = \operatorname{argmax}_{i \in 1..n} \sum_{j=1}^n |X_{i,:}WX_{j,:}^{\mathsf{T}}|$ and a vector $k^* = \operatorname{sign} X_{i,:}W$. The resulting constraint on $W(:)$ can be written as $k^*LW(:) \leq 1$ where $L_{j,:} = \xi^{(i^*j)}$, $i = 1..n$. This formulation allows us to use a general QP solver to compute $\tilde{W}$.

Note that batch fitted value iteration performs many regressions where the targets $y$ change from iteration to iteration, but the design matrix $X$ is fixed. Therefore we only need to solve the ECOLS optimization problem once for any given application of FVI, meaning the additional computational cost of ECOLS over OLS is not a major drawback.

## 6 Experimental results

In order to illustrate the behavior of ECOLS in different settings, we present four different empirical evaluations: one regression problem and three RL problems. In each of the RL settings, ECOLS with FVI converges, and the learned value function defines a good greedy policy.

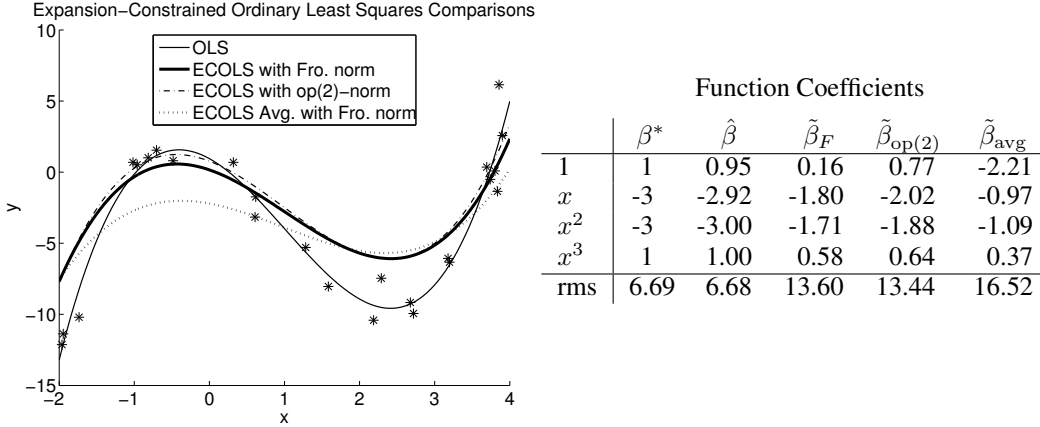

Figure 1: Example of OLS, ECOLS with $||XWX^\mathsf{T} - H||_F$, ECOLS with $||XWX^\mathsf{T} - H||_{\mathrm{op}(2)}$

**Regression**  The first is a simple regression setting, where we examine the behavior of ECOLS compared to OLS. To give a simple, pictorial rendition of the difference between OLS, ECOLS using the Frobenius, ECOLS using the $\mathrm{op}(2)$-norm, and an averager, we generated a dataset of $n = 25$ tuples $(x, y)$ as follows: $x \sim U(-2, 4)$, $y = 1 - 3x - 3x^2 + x^3 + \varepsilon$, $\varepsilon \sim N(0, 4)$. The design matrix $X$ had rows $X_{i,:} = [1, x_i, x_i^2, x_i^3]$. The ECOLS regression optimizing the Frobenius norm using CPLEX [12] took 0.36 seconds, whereas optimizing the $\mathrm{op}(2)$-norm using the cvx package [10] took 8.97 seconds on a 2 GHz Intel Core 2 Duo.

Figure 1 shows the regression curves produced by OLS and the two versions of ECOLS, along with the learned coefficients and root mean squared error of the predictions on the data. Neither of the ECOLS curves fit the data as well as OLS, as one would expect. Generally, their curves are smoother than the OLS fit, and predictions are on the whole shrunk toward zero. We also ran ECOLS with an additional positivity constraint on $X\tilde{W}X^\mathsf{T}$, effectively forcing the result to be an averager as described in Sect. 3. The result is smoother than either of the ECOLS regressors, with a higher RMS prediction error. Note the small difference between ECOLS using the Frobenius norm (dark black line) and using the $\mathrm{op}(2)$-norm (dashed line.) This is encouraging, as we have found that in larger datasets optimizing the $\mathrm{op}(2)$-norm is much slower and less reliable.

**Two-state example**  Our second example is a classic on-policy fitted value iteration problem that is known to diverge using OLS. It is perhaps the simplest example of FVI diverging, due to Tsitsiklis and Van Roy [22]. This is a deterministic on-policy example, or equivalently for our purposes, a problem with $|A| = 1$. There are three states $\{1, 2, 3\}$ with features $X = (1, 2, 0)^\mathsf{T}$, one action with $P_{1,2} = 1, P_{2,2} = 1 - \varepsilon, P_{2,3} = \varepsilon, P_{3,3} = 1$ and $P_{i,j} = 0$ elsewhere. The reward is $R = [0, 0, 0]^\mathsf{T}$ and the value function is $v^* = [0, 0, 0]^\mathsf{T}$. For $\gamma > 5/(6 - 4\varepsilon)$, FVI with OLS diverges for any starting point other than $v^*$. FVI with ECOLS always converges to $v^*$. If we change the reward to $R = [1, 1, 0]^\mathsf{T}$ and set $\gamma = 0.95$, $\varepsilon = 0.1$, we have $v^* = [7.55, 6.90, 0]$. FVI with OLS of course still diverges, whereas FVI with ECOLS converges to $\tilde{v} = [4.41, 8.82, 0]$. In this case, the approximation space is poor, and no linear method based on the features in $X$ can hope to perform well. Nonetheless, ECOLS converges to a $\hat{v}$ of at least the appropriate magnitude.

**Grid world**  Our third example is an off-policy value iteration problem which is known to diverge with OLS, due to Boyan and Moore [4]. In this example, there are effectively 441 discrete states, laid out in a $21 \times 21$ grid, and assigned an $(x, y)$ feature in $[0, 1]^2$ according to their position in the grid. There are four actions which deterministically move the agent up, down, left, or right by a distance of 0.05 in the feature space, and the reward is -0.5 everywhere except the corner state $(1, 1)$, where it is 0. The discount $\gamma$ is set to 1.0 so the optimal value function is $v^*(x, y) = -20 + 10x + 10y$.

Boyan and Moore define "lucky" convergence of FVI as the case where the policy induced by the learned value function is optimal, even if the learned value function itself does not accurately represent $v^*$. They found that with OLS and a design matrix $X_{i,:} = [1, x_i, y_i]$, they achieve lucky convergence. We replicated their result using FVI on 255 randomly sampled states plus the goal

state, and found that OLS converged[5] to $\hat{\beta} = [-515.89, 9.99, 9.99]$ after 10455 iterations. This value function induces a policy that attempts to increase $x$ and $y$, which is optimal. ECOLS on the other hand converged to $\tilde{\beta} = [-1.09, 0.030, 0.07]$ after 31 iterations, which also induces an optimal policy. In terms of learning correct value function coefficients, the OLS estimate gets 2 of the 3 almost exactly correct. In terms of estimating the value of states, OLS achieves an RMSE over all states of 10413.73, whereas ECOLS achieves an RMSE of 208.41.

In the same work, Boyan and Moore apply OLS with quadratic features $X_{i,:} = [1, x, y, x^2, y^2, xy]$, and find that FVI diverges. We found that ECOLS converges, with coefficients $[-0.80, -2.67, -2.78, 2.73, 2.91, 0.06]$. This is not "lucky", as the induced policy is only optimal for states in the upper-right half of the state space.

**Left-or-right world** Our fourth and last example is an off-policy value iteration problem with stochastic dynamics where OLS causes non-divergent but non-convergent behavior. To investigate properties of their tree-based Fitted Q-Iteration (FQI) methods, Ernst, Geurts, and Wehenkel define the "left-or-right" problem [5], an MDP with $S = [0, 10]$, and stochastic dynamics given by $s^{t+1} = s^t + a + \varepsilon$, where $\varepsilon \sim N(0, 1)$. Rewards are 0 for $s \in [0, 10]$, 100 for $s > 10$, and 50 for $s < 0$. All states outside $[0, 10]$ are terminal. The discount factor $\gamma$ is 0.75. In their formulation they use $A \in \{-2, 2\}$, which gives an optimal policy that is approximately $\pi^*(s) = \{2$ if $s > 2.5$, -2 otherwise$\}$. We examine a simpler scenario by choosing $A \in \{-4, 4\}$, so that $\pi^*(s) = 4$, i.e., it is optimal to always go right. Based on prior data [5], the optimal $Q$ functions for this type of problem appear to be smooth and non-linear, possibly with inflection points. Thus we use polynomial features[6] $X_{i,:} = [1, x, x^2, x^3]$ where $x = s/5 - 1$. As is common in FQI, we fit separate regressions to learn $Q(\cdot, 4)$ and $Q(\cdot, -4)$ at each iteration. We used 300 episodes worth of data generated by the uniform random policy for learning.

In this setting, OLS does not diverge, but neither does it converge: the parameter vector of each $Q$ function moves chaotically within some bounded region of $\mathbb{R}^4$. The optimal policy induced by the Q-functions is determined solely by zeroes of $Q(\cdot, 4) - Q(\cdot, -4)$, and in our experiments this function had at most one zero. Over 500 iterations of FQI with OLS, the cutpoint ranged from -7.77 to 14.04, resulting in policies ranging from "always go right" to "always go left.' FQI with ECOLS converged to a near-optimal policy $\tilde{\pi}(s) = \{4$ if $s > 1.81$, -4 otherwise$\}$. We determined by Monte Carlo rollouts that, averaged over a uniform initial state, the value of $\tilde{\pi}$ is 59.59, whereas the value of the optimal policy $\pi^*$ is 60.70. While the performance of the learned policy is very good, the estimate of the average value using the learned $Q$s, 28.75, is lower due to the shrinkage induced by ECOLS in the predicted state-action values.

## 7   Concluding Remarks

Divergence of FVI with OLS has been a long-standing problem in the RL literature. In this paper, we introduced ECOLS, which provides guaranteed convergence of FVI. We proved theoretical properties that show that in the minimax sense, ECOLS is optimal among possible linear approximations that guarantee such convergence. Our test problems confirm the convergence properties of ECOLS and also illustrate some of its properties. In particular, the empirical results illustrate the regularization effect of the $\operatorname{op}(\infty)$-norm constraint that tends to "shrink" predicted values toward zero. This is a further contribution of our paper: Our theoretical and empirical results indicate that this shrinkage is a necessary cost of guaranteeing convergence of FVI using linear models with a fixed set of features. This has important implications for the deployment of FVI with ECOLS. In some applications where accurate estimates of policy performance are required, this shrinkage may be problematic; addressing this problem is an interesting avenue for future research. In other applications where the goal is to identify a good, intuitively represented value function and policy ECOLS, is a useful new tool.

**Acknowledgements** We acknowledge support from Natural Sciences and Engineering Research Council of Canada (NSERC) and the National Institutes of Health (NIH) grants R01 MH080015 and P50 DA10075.

## Footnotes

[1] A noisy reward signal does not alter the analyses that follow, nor does dependence of the reward on action.

[2]The original definition of an averager was an operator of the form $y \mapsto Ay + b$ for a constant vector $b$. For this work we assume $b = 0$.

[3]The $c$ is a mathematical convenience; if $||z||_2$ were unbounded then the max would be unbounded and the problem ill-posed.

[4]One could in principle include a semi-definite constraint in the problem formulation, at an increased computational cost. (The problem is not a standard semi-definite program because the objective is not linear in the elements of $W$.) We have not imposed this constraint in our experiments and we have always found that the resulting $\tilde{W}$ is positive semi-definite. We conjecture that $\tilde{W}$ is always positive semi-definite.

[5]Convergence criterion was $||\beta^{\mathrm{iter}+1} - \beta^{\mathrm{iter}}|| \leq 10^{-5}$. All starts were from $\beta = 0$.

[6]The re-scaling of $s$ is for numerical stability.

# References

[1] A. Antos, R. Munos, and Cs. Szepesvári. Fitted Q-iteration in continuous action-space MDPs. In *Advances in Neural Information Processing Systems 20*, pages 9–16. MIT Press, 2008.

[2] L. Baird. Residual Algorithms: Reinforcement Learning with Function Approximation. In A. Prieditis and S. Russell, editors, *Proceedings of the 25th International Conference on Machine Learning*, pages 30–37. Morgan Kaufmann, 1995.

[3] D. Bertsekas. *Dynamic Programming and Optimal Control*. Athena Scientific, 2007.

[4] J. Boyan and A. W. Moore. Generalization in reinforcement learning: Safely approximating the value function. In *Advances in neural information processing systems*, pages 369–376, 1995.

[5] D. Ernst, P. Geurts, and L. Wehenkel. Tree-Based Batch Mode Reinforcement Learning. *Journal of Machine Learning Research*, 6:503–556, 2005.

[6] A. M. Farahmand, M. Ghavamzadeh, Cs. Szepesvári, and S. Mannor. Regularized fitted Q-iteration for planning in continuous-space Markovian decision problems. In *American Control Conference*, pages 725–730, 2009.

[7] R. Fonteneau. *Contributions to Batch Mode Reinforcement Learning*. PhD thesis, University of Liege, 2011.

[8] G. J. Gordon. *Approximate Solutions to Markov Decision Processes*. PhD thesis, Carnegie Mellon University, 1999.

[9] M. Grant and S. Boyd. CVX: Matlab software for disciplined convex programming, version 1.21. http://cvxr.com/cvx, Apr. 2011.

[10] M. C. Grant. Disciplined convex programming and the cvx modeling framework. *Information Systems Journal*, 2006.

[11] A. Guez, R. D. Vincent, M. Avoli, and J. Pineau. Adaptive treatment of epilepsy via batch-mode reinforcement learning. In D. Fox and C. P. Gomes, editors, *Innovative Applications of Artificial Intelligence*, pages 1671–1678, 2008.

[12] IBM. IBM ILOG CPLEX Optimization Studio V12.2, 2011.

[13] S. Kalyanakrishnan and P. Stone. Batch reinforcement learning in a complex domain. In *Proceedings of the 6th international joint conference on Autonomous agents and multiagent systems AAMAS 07*, 2007.

[14] R. Munos and Cs. Szepesvári. Finite time bounds for fitted value iteration. *Journal of Machine Learning Research*, 9:815–857, 2008.

[15] D. Ormoneit and S. Sen. Kernel-based reinforcement learning. *Machine learning*, 49(2):161–178, 2002.

[16] M. Riedmiller. Neural fitted Q iteration-first experiences with a data efficient neural reinforcement learning method. In *ECML 2005*, pages 317–328. Springer, 2005.

[17] J. Rust. Using randomization to break the curse of dimensionality. *Econometrica*, 65(3):pp. 487–516, 1997.

[18] G. A. F. Seber. *A MATRIX HANDBOOK FOR STATISTICIANS*. Wiley, 2007.

[19] S. M. Shortreed, E. Laber, D. J. Lizotte, T. S. Stroup, J. Pineau, and S. A. Murphy. Informing sequential clinical decision-making through reinforcement learning : an empirical study. *Machine Learning*, 2010.

[20] S. Siddiqi, B. Boots, and G. Gordon. A Constraint Generation Approach to Learning Stable Linear Dynamical Systems. In *Advances in Neural Information Processing Systems 20*, pages 1329–1336. MIT Press, 2008.

[21] Cs. Szepesvári. *Algorithms for Reinforcement Learning*. Morgan and Claypool, 2010.

[22] J. N. Tsitsiklis and B. van Roy. An analysis of temporal-difference learning with function approximation. *IEEE Transactions on Automatic Control*, 42(5):674–690, 1997.

